# The doubly balanced network of spiking neurons: a memory model with high capacity

**Yuval Aviel**[*]
Interdisciplinary Center for Neural Computation
Hebrew University
Jerusalem, Israel 91904
*aviel@cc.huji.ac.il*

**David Horn**
School of Physics
Tel Aviv University
Tel Aviv, Israel 69978
*horn@post.tau.ac.il*

**Moshe Abeles**
Interdisciplinary Center for Neural Computation
Hebrew University
Jerusalem, Israel 91904
*abeles@vms.huji.ac.il*

## Abstract

A balanced network leads to contradictory constraints on memory models, as exemplified in previous work on accommodation of synfire chains. Here we show that these constraints can be overcome by introducing a 'shadow' inhibitory pattern for each excitatory pattern of the model. This is interpreted as a double-balance principle, whereby there exists both global balance between average excitatory and inhibitory currents and local balance between the currents carrying coherent activity at any given time frame. This principle can be applied to networks with Hebbian cell assemblies, leading to a high capacity of the associative memory. The number of possible patterns is limited by a combinatorial constraint that turns out to be *P=0.06N* within the specific model that we employ. This limit is reached by the Hebbian cell assembly network. To the best of our knowledge this is the first time that such high memory capacities are demonstrated in the asynchronous state of models of spiking neurons.

## 1 Introduction

Numerous studies analyze the different phases of unstructured networks of spiking neurons [1, 2]. These networks with random connectivity possess a phase of asynchronous activity, the asynchronous state (AS), which is the most interesting one from the biological perspective, since it is similar to physiological data. Unstructured networks, however, do not hold information in their connectivity matrix, and therefore do not store memories.

Binary networks with ordered connectivity matrices, or structured networks, and their ability to store and retrieve memories, have been extensively studied in the past [3-8]. Applicability of these results to biologically plausible neuronal models is questionable. In particular, models of spiking neurons are known to have modes of synchronous global oscillations. Avoiding such modes, and staying in an AS, is a major constraint on networks of spiking neurons that is absent in most binary neural networks. As we will show below, it is this constraint that imposes a limit on capacity in our model. Existing associative memory models of spiking neurons have not strived for maximal pattern capacity [3, 4, 8].

Here, using an integrate-and-fire model, we embed structured synaptic connections in an otherwise unstructured network and study the capacity limit of the system. The system is therefore macroscopically unstructured, but microscopically structured. The unstructured network model is based on Brunel's [1] balanced network of integrate-and-fire neurons. In his model, the network possesses different phases, one of which is the AS. We replace his unstructured excitatory connectivity by a semi-structured one, including a super-position of either synfire chains or Hebbian cell assemblies.

The existence of a stable AS is a fundamental prerequisite of the system. There are two reasons for that: First, physiological measurements of cortical tissues reveal an irregular neuronal activity and an asynchronous population activity. These findings match the properties of the AS. Second, in term of information content, the entropy of the system is the highest when firing probability is uniformly distributed, as in an AS. In general, embedding one or two patterns will not destabilize the AS. Increasing the number of embedded patterns, however, will eventually destabilize the AS, leading to global oscillations.

In previous work [9], we have demonstrated that the cause of AS instability is correlations between neurons that result from the presence of structure in the network. The patterns, be it Hebbian cell assemblies (HCA) or pools occurring in synfire chains (SFC), have an important characteristic: neurons that are members of the same pattern (or pool) share a large portion of their inputs. This common input correlates neuronal activities both when a pattern is activated and when both neurons are influenced by random activity. If too many patterns are embedded in the network, too many neurons become correlated due to common inputs, leading to globally synchronized deviations from mean activity.

A qualitative understanding of this state of affairs is provided by a simple model of a threshold linear pair of neurons that receive $n$ excitatory common, and correlated, inputs, and $K$-$n$ excitatory, as well as $K$ inhibitory, non-common uncorrelated inputs. Thinking of these neurons as belonging to a pattern or a pool within a network, we can obtain an interesting self-consistent result by assuming the correlation of the pair of neurons to be also the correlation in their common correlated input (as is likely to be the case in a network loaded with HCA or SFC).

We find then [9] that there exists a critical pattern size, $n_c$, below which correlations decay but above which correlations are amplified. Furthermore, the following scaling was found to exist

$$(1) \qquad\qquad n_c = r_c \sqrt{K} .$$

Implications of this model for the whole network are that: (i) $r_c$ is independent of $N$, the size of the network, (ii) below $n_c$ the AS is stable, and (iii) above $n_c$ the AS is unstable.

Using extensive computer simulations we were able [9] to validate all these predictions. In addition, keeping $n<n_c$, we were able to observe traveling synfire waves on top of global asynchronous activity.

The pattern's size $n$ is also limited from below, $n> n_{min}$, by the requirement that $n$ excitatory post-synaptic potentials (PSPs), on average, drive a neuron across its threshold. Since $N>K$ and typically $N>>K$, together with Eq. (1) it follows that $N >> \left(n_{\min} / r_c\right)^2$. Hence $r_c$ and $n_{min}$ set the lower bound of the network's size, above which it is possible to embed a reasonable number of patterns in the network without losing the AS. In this paper we propose a solution that enables small $n_{min}$ and large $r$ values, which in turn enables embedding a large number of patterns in much smaller networks. This is made possible by the doubly-balanced construction to be outlined below.

## 2   The double-balance principle

Counteracting the excitatory correlations with inhibitory ones is the principle that will allow us to solve the problem. Since we deal with balanced networks, in which the mean excitatory input is balanced by an inhibitory one, we note that this principle imposes a second type of balancing condition, hence we refer to it as the *double- balance principle*.

In the following, we apply this principle by introducing synaptic connections between any excitatory pattern and its randomly chosen inhibitory pattern. These inhibitory patterns, which we call *shadow patterns*, are activated after the excitatory patterns fire, but have no special in-pattern connectivity or structured projections onto other patterns. The premise is that correlations evolved in the excitatory patterns will elicit correlated inhibitory activity, thus balancing the network's average correlation level. The size of the shadow pattern has to be small enough, so that the global network activity will not be quenched, yet large enough, so that the excitatory correlation will be counteracted. A balanced network that is embedded with patterns and their shadow patterns will be referred to as a doubly balanced network (DBN), to be contrasted with the singly balanced network (SBN) where shadow patterns are absent.

## 3   Application of the double balance principle.

### 3.1   The Network

We model neuronal activity with the Integrate and Fire [10] model. All neurons have the same parameters: $\tau = 10ms$, $\tau_{ref} = 2.5ms$, $C=250pF$. PSPs are modeled by a delta function with fixed delay. The number of synapses on a neuron is fixed and set to $K_E$ excitatory synapses from the local network, $K_E$ excitatory synapses from external sources and $K_I$ inhibitory synapses from the local network. See Aviel et al [9] for details. All synapses of each group will be given fixed values. It is allowed for one pre-synaptic neuron to make more than one connection to one post-synaptic neuron. The network possesses $N_E$ excitatory neurons and $N_I \equiv \gamma N_E$ inhibitory neurons. Connectivity is sparse, $K_E/N_E = K_I/N_I = \varepsilon$, (we use $\varepsilon = 0.1$). A Poisson process with rate $v_{ext}$=10Hz models the external source. If a

neuron of population $y$ innervates a neuron of population $x$ its synaptic strength $J_{xy}$ is defined as

$$J_{xE} \equiv J_0 \Big/ \sqrt{K_E} \, , \; J_{xI} \equiv -gJ_0 \Big/ \sqrt{K_I}$$

with $J_0$=10, and $g$=5. Note that $J_{xI} = -\frac{g}{\sqrt{\gamma}} J_{xE}$, hence $\frac{g}{\sqrt{\gamma}}$ controls the balance between the two populations.

Within an HCA pattern the neurons have high connection probability with one another. Here it is achieved by requiring $L$ of the synapses of a neuron in the excitatory pattern to originate from within the pattern. Similarly, a neuron in the inhibitory shadow pattern dedicates $L$ of its synapses to the associated excitatory pattern. In a SFC, each neuron in an excitatory pool is fed by $L$ neurons from the previous pool. This forms a feed forward connectivity. In addition, when shadow pools are present, each neuron in a shadow pool is fed by $L$ neurons from its associated excitatory pool.

In both cases $L = C_L \sqrt{K_E}$ , with $C_L$=2.5. The size of the excitatory patterns (i.e. the number of neurons participating in a pattern) or pools, $n_E$, is also chosen to be proportional to $\sqrt{K_E}$ (see Aviel et al. 2003 [9]), $n_E \equiv C_n \sqrt{K_E}$ , where $C_n$ varies. This is a suitable choice, because of the behavior of the critical $n_c$ of Eq. (1), and is needed for the meaningful memory activity (of the HCA or SFC) to overcome synaptic noise.

The size of a shadow pattern is defined as $n_I \equiv \tilde{d} n_E$ . This leads to the factor d, representing the relative strength of inhibitory and excitatory currents, due to a pattern or pool, affecting a neuron that is connected to both:

$$(2) \qquad d \equiv \frac{-J_{xI} n_I}{J_{xE} n_E} = \frac{gJ_0 \sqrt{K_E} \, \tilde{d}}{J_0 \sqrt{K_I}} = \frac{g\tilde{d}}{\sqrt{\gamma}} \, .$$

Thus it fixes $n_I = d \left( \sqrt{\gamma} \big/ g \right) n_E$. In the simulations reported below $d$ varied between 1 and 3.

Wiring the network is done in two stages, first all excitatory patterns are wired, and then random connections are added, complying with the fixed number of synapses.

A volley of $w$ spikes, normally distributed over time with width of 1ms, is used to ignite a memory pattern. In the case of SFC, the first pool is ignited, and under the right conditions the volley propagates along the chain without fading away and without destabilizing the AS.

## 3.2  Results

First we show that the AS remains stable when embedding HCAs in a small DBN, whereas global oscillations take place if embedding is done without shadow pools. Figure 1 displays clearly the sustained activity of an HCA in the DBN.

The same principle also enables embedding of SFCs in a small network. This is to be contrasted with the conclusions drawn in Aviel et al [9], where it was shown that otherwise very large networks are necessary to reach this goal.

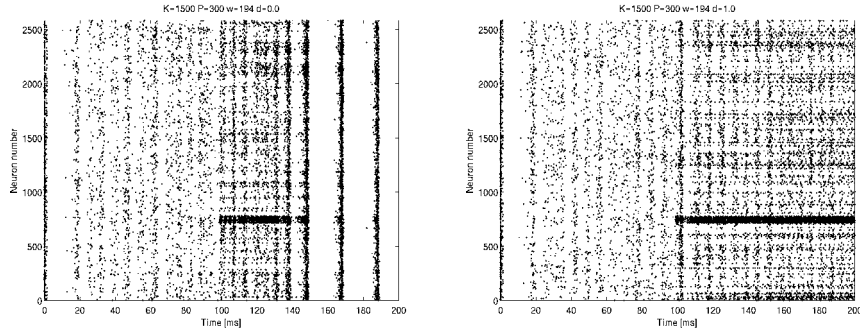

Figure 1: HCAs are embedded in a balanced network without (left) and with (right) shadow patterns. $P$=300 HCAs of size $n_E$=194 excitatory neurons were embedded in a network of $N_E$=15,000 excitatory neurons. The eleventh pattern is externally ignited at time t=100ms. A raster plot of 200ms is displayed. Without shadow patterns the network exhibits global oscillations, but with shadow patterns the network exhibits only minute oscillations, enabling the activity of the ignited pattern to be sustained. The size of the shadow patterns is set according to Eq. (2) with $d$=1. Neurons that participate in more than one HCA may appear more than once on the raster plot, whose y-axis is ordered according to HCAs, and represents every second neuron in each pattern.

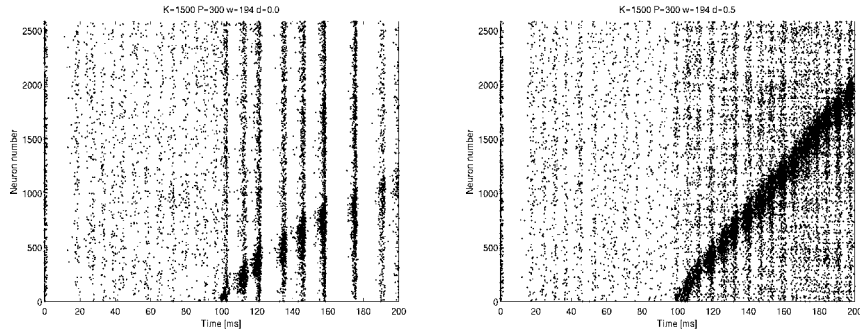

Figure 2: SFCs embedded in a balanced network without (left) and with (right) shadow patterns. The first pool is externally ignited at time t=100ms. $d$=0.5. The rest of the parameters are as in Figure 1. Here again, without shadow pools, the network exhibits global oscillations, but with shadow pools it has only minute oscillation, enabling a stable propagation of the synfire wave.

### 3.3 Maximum Capacity

In this section we show that, within our DBN, it is the fixed number of synapses (rather than dynamical constraints) that dictates the maximal number of patterns or pools $P$ that may be loaded onto the network. Let us start by noting that a neuron of population $x$ (E or I) can participate in at most $m \equiv \lfloor K_E / L \rfloor$ patterns, hence $N_x m$

sets an upper bound on the number of neurons that participate in all patterns: $n_x P \leq m \cdot N_x$. Next, defining $\alpha_x \equiv \dfrac{P}{N_x}$, we find that

$$(3) \qquad \alpha_x \leq \frac{m}{n_x} = \frac{\left\lfloor K_E / C_L \sqrt{K_E} \right\rfloor}{n_x}$$

To leading order in $N_E$ this turns into

$$(4) \qquad \alpha_x N_x = \frac{\left\lfloor K_E / C_L \sqrt{K_E} \right\rfloor}{D_x C_n \sqrt{K_E}} N_E = \left( C_n C_L D_x \right)^{-1} N_E - O\left( \sqrt{N_E} \right)$$

where $D_x \equiv d / \left( g \sqrt{\gamma} \right)$ if $x=I$, or 1 for $x=E$.

Thus we conclude that synaptic combinatorial considerations lead to a maximal number of patterns $P$. If $D_I < 1$, including the case $D_I = 0$ of the SBN, the excitatory neurons determine the limit to be $P = \left( C_n C_L \right)^{-1} N_E$. If, as is the case in our DBN, $D_I > 1$, then $\gamma \alpha_I < \alpha_E$ and the inhibitory neurons set the maximum value to $P = \left( C_n C_L D_I \right)^{-1} N_E$.

For example, setting $C_n=3.5$, $C_L=2.4$, $g=3$ and $d=3$, in Eq. (4), we get $P=0.06 N_E$. In Figure 3 we use these parameters. The capacity of a DBN is compared to that of an SBN for different network sizes. The maximal load is defined by the presence of global oscillation strong enough to prohibit sustained activity of patterns. The DBN reaches the combinatorial limit, whereas the SBN does not increase with $N$ and obviously does not reach its combinatorial limit.

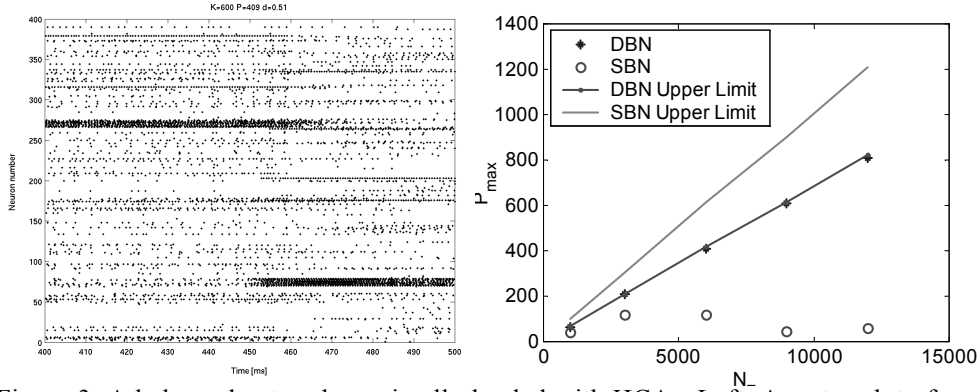

Figure 3: A balanced network maximally loaded with HCAs. Left: A raster plot of a maximally loaded DBN. $P=408$, $N_E=6,000$. At time t=450ms, the seventh pattern is ignited for a duration of 10ms, leading to termination of another pattern's activity (upper stripe) and to sustained activity of the ignited pattern (lower stripe). Right: $P(N_E)$ as inferred from simulations of a SBN ("o") and of a DBN ("*"). The DBN realizes the combinatorial limit (dashed line) whereas the SBN does not realize its limit (solid line). From this comparison it is clear that DBN is superior to the SBN in terms of network capacity.

The simulations displayed in Figure 3 show that in the DBN the combinatorial $P$ is indeed realized, and the capacity of this DBN grows like $0.06N_E$. In the SBN, dynamic interference prevents reaching the combinatorial limit.

We have tried, in many ways, to increase the capacity of SBN. Recently, we have discovered [11] that only if the external rates are appropriately scaled, then SBN capacity can be linear with $N_E$ with a pre-factor $\alpha$ almost as high as that of a DBN. Although under these conditions SBNs can have large capacity, we emphasize that DBNs posses a clear advantage. Their structure guarantees high capacity under more general conditions.

## 4  Discussion

In this paper we study memory patterns embedded in a balanced network of spiking neurons. In particular, we focus on the maximal capacity of Hebbian cell assemblies. Requiring stability of the asynchronous state of the network, that serves as the background for memory activity, and further assuming that the neuronal spiking process is noise-driven, we show that naively applying Hebb's architecture leads to global oscillations. We propose the double-balance principle as the solution to this problem. This double-balance is obtained by introducing shadow patterns, i.e. inhibitory patterns that are associated with the excitatory ones and fed by them, but do not have specific connectivity other than that.

The maximal load of our system is determined in terms of the available synaptic resources, and is proportional to the size of the excitatory population, $N_E$. For the parameters used here it turns out to be $P=0.06N_E$. This limit was estimated by a combinatorial argument of synaptic availability, and shown to be realized by simulations.

Synfire chains were also studied. DBNs allow for their embedding in relatively small networks, as shown in Figure 2. Previous studies have shown that their embedding in balanced networks without shadow pools require network sizes larger by an order of magnitude [9]. The capacity $P$ of a SFC is defined, in analogy with the HCA case, as the number of pools embedded in the network. In this case we cannot realize the theoretical limit in simulations. We believe that the feed-forward structure of the SFC, which is absent in HCA, introduces further dynamical interference. The feed-forward structure can amplify correlations and firing rates more efficiently than the feedback structure within patterns of the HCA. Thus a network embedded with SFCs may be more sensitive to spontaneously evolved correlations than a network embedded with HCAs.

It is interesting to note that the addition of shadow patterns has an analogy in the Hopfield model [5], where neurons in a pattern have both excitatory and inhibitory couplings with the rest of the network. One may claim that the architecture proposed here recovers the same effect via the shadow patterns. Accommodating the Hopfield model in networks of spiking neurons was tried before [3, 4] without specific emphasis on the question of capacity. In Gerstner and van Hemenn [4] the synaptic matrix is constructed in the same way as in the Hopfield model, i.e. neurons can have excitatory and inhibitory synapses. In [3, 8] the synaptic bonds of the Hopfield model were replaced by strong excitatory connections within a pattern, and weak excitatory connections among neurons in a patterns and those outside the pattern. While the different types of connection are of different magnitude, they are all excitatory. In contrast, here, excitation exists within a pattern as well as outside it, but the pattern has a well-defined inhibitory effect on the rest of the network, mediated by the shadow pattern. The resulting inhibitory correlated currents cancel the excitatory correlated input. Since the firing process in a BN is driven by

fluctuations, it seems that negating excitatory correlations by inhibitory ones is more akin to Hopfield's construction in a network of two populations.

Hertz [12] has argued that a capacity limit obtained in a network of integrate-and-fire neurons should be multiplied by $\tau/2$ to compare it with a network of binary neurons. Hence the $\alpha = 0.12$ obtained here, is equivalent to $\alpha = 0.6$ in a binary model. It is not surprising that the last number is higher than 0.14, the limit of the original Hopfield model, since our model is sparse, as, e.g. the Tsodyks-Feigelman [7] model, where larger capacities were achieved.

Finally, let us point out again that whereas only DBNs can reach the combinatorial capacity limit under the conditions specified in this paper, we have recently discovered [11] that SBN can also reach this limit if additional scaling conditions are imposed on the input. The largest capacities that we obtained under these conditions were of order 0.1.

## Acknowledgments

This work was supported in part by grants from GIF.

## References

1. Brunel, N., *Dynamics of sparsely connected networks of excitatory and inhibitory spiking neurons.* J Comput Neurosci, 2000. **8**(3): p. 183-208.

2. van Vreeswijk, C. and H. Sompolinsky, *Chaotic balanced state in a model of cortical circuits.* Neural Comput, 1998. **10**(6): p. 1321-71.

3. Amit, D., J and N. Brunel, *Dynamics of a recurrent network of spiking neurons before and following learning.* Network, 1997. **8**: p. 373.

4. Gerstner, W. and L. van Hemmen, *Associative memory in a network of 'spiking' neurons.* Network, 1992. **3**: p. 139-164.

5. Hopfield, J.J., *Neural networks and physical systems woth emergant collective computational abilities.* PNAS, 1982. **79**: p. 2554-58.

6. Willshaw, D.J., O.P. Buneman, and H.C. Longuet-Higgins, *Non-holographic associative memory.* Nature (London), 1969. **222**: p. 960-962.

7. Tsodyks, M.V. and M.V. Feigelman, *The enhanced storage capacity in neural networks with low activity level.* Europhys. Let., 1988. **6**(2): p. 101.

8. Brunel, N. and X.-J. Wang, *Effects of neuromodulation in a cortical network model of object working memory dominated by recurrent inhibition.* J. of Computational Neuroscience, 2001. **11**: p. 63-85.

9. Aviel, Y., et al., *On embedding synfire chains in a balanced network.* Neural Computation, 2003. **15**(6): p. 1321-1340.

10. Tuckwell, H.C., *Introduction to theoretical neurobiology.* 1988, Cambridge: Cambridge University Press.

11. Aviel, Y., D. Horn, and M. Abeles, *Memory Capacity of Balanced Networks.* 2003: Submitted.

12. Hertz, J.A., *Modeling synfire networks*, in *Neuronal Information processing - From Biological Data to Modelling and Application*, G. Burdet, P. Combe, and O. Parodi, Editors. 1999.
